# Reconciling Real Scores with Binary Comparisons: A Unified Logistic Model for Ranking

**Nir Ailon**
Google Research NY
111 8th Ave, 4th FL New York NY 10011 `nailon@gmail.com`

## Abstract

The problem of ranking arises ubiquitously in almost every aspect of life, and in particular in Machine Learning/Information Retrieval. A statistical model for ranking predicts how humans rank subsets $V$ of some universe $U$. In this work we define a statistical model for ranking that satisfies certain desirable properties.

The model automatically gives rise to a logistic regression based approach to learning how to rank, for which the score and comparison based approaches are dual views. This offers a new generative approach to ranking which can be used for IR.

There are two main contexts for this work. The first is the theory of econometrics and study of statistical models explaining human choice of alternatives. In this context, we will compare our model with other well known models. The second context is the problem of ranking in machine learning, usually arising in the context of information retrieval. Here, much work has been done in the discriminative setting, where different heuristics are used to define ranking risk functions.

Our model is built rigorously and axiomatically based on very simple desirable properties defined locally for comparisons, and automatically implies the existence of a global score function serving as a natural model parameter which can be efficiently fitted to pairwise comparison judgment data by solving a convex optimization problem.

## 1 Introduction

Ranking is an important task in information sciences. The most notable application is information retrieval (IR), where it is crucial to return results in a sorted order for the querier. The subject of preference and ranking has been thoroughly studied in the context of statistics and econometric theory [8, 7, 29, 36, 34, 31], combinatorial optimization [26, 37, 20, 3, 4, 14] and machine learning [6, 9, 33, 21, 19, 35, 23, 22, 25, 16, 17, 1, 13, 15, 28, 18].

Recently Ailon and Mehryar [5] following Balcan et al [9] have made significant progress in reducing the task of learning ranking to the binary classification problem of learning preferences. This comparison based approach is in contrast with a *score* based approach which tries to regress to a score function on the elements we wish to rank, and sort the elements based on this score as a final step.

The difference between the score based and comparison approaches is an example of "local vs. global" views: A comparison is local (how do two elements compare with each other), and a score is global (how do we embed the universe on a scale). The score based approach seems reasonable in cases where the score can be defined naturally in terms of measurable utility. In some real world scenarios, either (i) an interpretable score is difficult to define (e.g. a relevance score in information retrieval) and (ii) an interpretable score is easy to define (e.g. how much a random person is willing

to pay for product X in some population) but learning the score is difficult due to noisy or costly label acquisition for scores on individual points [7].

A well known phenomenon in the psychological study of human choice seems to potentially offer an elegant solution to the above difficulties: Human response to comparison questions is more stable in the sense that it is not easily affected by irrelevant alternatives. This phenomenon makes acquisition of comparison labels for learning tasks more appealing, but raises the question of how to go back and fit a latent score function that explains the comparisons. Moreover, the score parameter fitting must be computationally efficient. Much effort has been recently put in this subject from a machine learning perspective [6, 9, 33, 21, 19, 35, 23, 22, 25, 16, 17, 1, 13, 15, 28, 18].

## 2 Ranking in Context

The study of ranking alternatives has not been introduced by ML/IR, and has been studied throughly from the early years of the 20th century in the context of statistics and econometrics. We mention work in ML/IR by Lebanon and Lafferty [27] and Cao et al. [12] who also draw from the classic work for information retrieval purposes.

ML/IR is usually interested in the question of how a *machine* should correctly rank alternatives based on experience from human feedback, whereas in statistics and econometrics the focus is on the question of how a *human* chooses from alternatives (for the purpose of e.g. effective marketing or policy making). Therefore, there are notable differences between the modern and classic foci. Notwithstanding these differences, the classic foci is relevant to modern applications, and vice versa. For example, any attempt to correctly choose from a set (predominantly asked in the classic context) can be converted into a ranking algorithm by repeatedly choosing and removing from the set.

**Definition 2.1** *A ranking model for $U$ is a function $\mathcal{D}$ mapping any finite subset $V \subseteq U$ to a distribution on rankings of $V$. In other words, $\mathcal{D}(V)$ is a probability distribution on the $|V|!$ possible orderings of $V$.*

A Thurstonian model for ranking (so named after L. Thurstone [36]) is one in which an independent random real valued variable $Z_v$ is associated with each $v \in V$, and the ranking is obtained by sorting the elements if $V$ in decreasing order (assuming the value represents utility). Often the distributions governing the $Z_v$'s are members of a parametric family, with a location parameter representing an intrinsic "value". The source of variability in $Z_v$ is beyond the scope of this work. This model is related to the more general random utility model (RUM) approach studied in econometrics.

A purely comparison based model is due to Babington and Smith: The parameter of the model is a matrix $\{p_{uv}\}_{u,v \in U}$. Given items $u, v$, a subject would prefer $u$ over $v$ with probability $p_{uv} = 1 - p_{vu}$. Given a subset $V$, the subject flips a corresponding biased coin independently to decide on the preference of all pairs $u, v \in V$, and repeats the process until the set of preferences is transitive. This model is unwieldy in full generality, and more succinct representations were proposed. Mallows [30] following Bradley and Terry [11] proposed to take $p_{uv}$ as $\alpha(u)/(\alpha(u) + \alpha(v))$, where the $\alpha(v)$'s are constants attached to each element. Note that the marginal probability of $u$ being preferred over $v$ in the context of a set $V \supset \{u, v\}$ in the Babington-Smith model is in general not $p_{uv}$, even in Mallows's special case.

In distance based models it is assumed that there is a "modal" ranking of the set $V$, and the probability of any ranking decreases with its distance from the mode. Several definitions of distances between permutations. Often the probability density itself is defined as an exponential model. We refer the reader to [31] for in depth analysis of such models.

**The Plackett-Luce model.** The classic model most related to this work is Plackett and Luce's [29, 34] multistage model for ranking. Each element $v \in U$ has an assigned "value" parameter $\alpha(v)$. At each stage a choice is made. Given a set $V$, item $u \in V$ wins with probability $\alpha(u)/\sum_{v \in V} \alpha(v)$.[1] The winner is removed from $V$ and the process is repeated for the remaining elements, until a ranking is obtained. Yellott [38] made the surprising observation that the Luce-Plackett model is exactly Thurstone's model where the $Z_u$'s are translated Gumbel (doubly-exponential) distributed

variables. The underlying winner choice model satisfies Luce's choice axiom [29] which, roughly speaking, stipulates that the probability of an element $u$ winning in $V$ is the same as the product of the probability of the winner contained in $V' \subseteq V$ and the probability of $u$ winning in $V'$. It turns out that this axiom (often used as criticism of the model) implies the underlying choice function of the Plackett-Luce model.

An interesting property of Plackett-Luce for our purpose is that it is *asymmetric* in the sense that it is winner-centric and not loser-centric. The model cannot explain both ranking by successive loser choice and successive winner choice simultaneously unless it is trivial (this point was noticed by McCullagh [32]). It is clear however that breaking down the process of ranking by humans to an iterated choice of winners ignores the process of elimination (placing alternatives at the bottom of the list). In the following sections we propose a new *symmetric* model for ranking, in which the basic discrete task is a comparison of pairs of elements, and not choice of an element from arbitrarily large sets (as in Plackett-Luce).

## 3 An Axiomatic Approach for Defining a Pairwise-Stable Model for Ranking

For a ranking $\pi$ of some subset $V \subseteq U$, we use the notation $u \prec_\pi v$ to denote that $u$ precedes[2] $v$ according to $\pi$. We let $\pi(v) \in \{1, \dots, n\}$ denote the rank of $v \in V$, where lower numbers designate precedence (hence $u \prec_\pi v$ if $\pi(u) < \pi(v)$). The inverse $\pi^{-1}(i)$ is the unique element $v$ of $V$ with $\pi(v) = i$. We overload notation and let $\pi(u, v)$ denote the indicator variable taking the value of 1 if $u \prec v$ and 0 otherwise.

**Definition 3.1** *A ranking model $\mathcal{D}$ for $U$ satisfies* pairwise stability *if for any $u, v \in U$ and for any $V_1, V_2 \supseteq \{u, v\}$, $\Pr_{\pi \sim \mathcal{D}(V_1)}[u \prec_\pi v] = \Pr_{\pi \sim \mathcal{D}(V_2)}[u \prec v]$.*

Pairwise stability means that the preference (or comparison) of $u, v$ is statistically independent of the context (subset) they are ranked in. Note that Plackett-Luce is pairwise stable (this follows from the fact that the model is Thurstonian) but Babington-Smith/Mallows is not. If a ranking model $\mathcal{D}$ satisfies pairwise stability, then the probability $\Pr_\mathcal{D}[u \prec v]$ is naturally defined and equals $\Pr_{\pi \sim \mathcal{D}(V)}[u \prec_\pi v]$ for any $V \supseteq \{u, v\}$.

Pairwise stability is a weak property which permits a very wide family of ranking distributions. In particular, if the universe $U$ is a finite set then any distribution $\Pi$ on rankings on the entire universe $U$ gives rise to a model $\mathcal{D}_\Pi$ with $\mathcal{D}_\Pi(V)$ defined as the restriction of $\Pi$ to $V$. This model clearly satisfies pairwise stability but does not have a succint description and hence undesirable.

We strengthen the conditions on our model by considering triplets of elements. Assume that a model $\mathcal{D}$ satisfies pairwise stability. Fix three elements $u, v, w$. Consider a process in which we randomly and independently decide how $u$ and $w$ should compare with $v$. What would be the induced distribution on the order of $u$ and $w$, conditioned on them being placed on opposite sides of $v$? If we sample from the distributions $\mathcal{D}(\{u, v\})$ and $\mathcal{D}(\{v, w\})$ to independently decide how to compare $u$ with $v$ and $w$ with $v$ (respectively), then we get

$$\Pr[u \prec w \mid (u \prec v \prec w) \vee (w \prec v \prec u)] =$$
$$\frac{\Pr_\mathcal{D}[u \prec v] \Pr_\mathcal{D}[v \prec w]}{\Pr_\mathcal{D}[u \prec v] \Pr_\mathcal{D}[v \prec w] + \Pr_\mathcal{D}[w \prec v] \Pr_\mathcal{D}[v \prec u]}.$$

What happens if we force this to equal $\Pr_\mathcal{D}[u \prec w]$? In words, this would mean that the comparison of $u$ with $w$ conditioned on the comparison being determined by pivoting around $v$ is distributed like $\mathcal{D}(\{u, w\})$. We write this desired property as follows (the second line follows from the first):

$$\Pr_{\mathcal{D}}[u \prec w] = \frac{\Pr_D[u \prec v]\Pr_{\mathcal{D}}[v \prec w]}{\Pr_D[u \prec v]\Pr_{\mathcal{D}}[v \prec w] + \Pr_{\mathcal{D}}[w \prec v]\Pr_{\mathcal{D}}[v \prec u]}$$

$$\Pr_{\mathcal{D}}[w \prec u] = \frac{\Pr_D[w \prec v]\Pr_{\mathcal{D}}[v \prec u]}{\Pr_D[w \prec v]\Pr_{\mathcal{D}}[v \prec u] + \Pr_{\mathcal{D}}[u \prec v]\Pr_{\mathcal{D}}[v \prec w]} \ . \tag{1}$$

**Definition 3.2** *Assume $\mathcal{D}$ is a ranking model for $U$ satisfying pairwise stability. For a pair $u, w \in U$ and another element $v \in U$ we say that $u$ and $w$ satisfy the pivot condition with respect to $v$ if (1) holds.*

Dividing the two desired equalities in (1), we get (assuming the ratio exists):

$$\frac{\Pr_{\mathcal{D}}[u \prec w]}{\Pr_{\mathcal{D}}[w \prec u]} = \frac{\Pr_{\mathcal{D}}[u \prec v]\Pr_{\mathcal{D}}[v \prec w]}{\Pr_{\mathcal{D}}[w \prec v]\Pr_{\mathcal{D}}[v \prec u]} \ . \tag{2}$$

If we denote by $\Delta_{\mathcal{D}}(a, b)$ the "comparison logit[3]": $\Delta_{\mathcal{D}}(a, b) = \log(\Pr_{\mathcal{D}}[a \prec b]/\Pr_{\mathcal{D}}[b \prec a])$, then (2) implies $\Delta_{\mathcal{D}}(u, v) + \Delta_{\mathcal{D}}(v, w) + \Delta_{\mathcal{D}}(w, u) = 0$. This in turn implies that there exist numbers $s_1, s_2, s_3$ such that $\Delta(u, v) = s_1 - s_2$, $\Delta(v, w) = s_2 - s_3$ and $\Delta(w, u) = s_1 - s_3$. These numbers, defined up to any additive constant, should be called (additive) *scores*. We will see in what follows that the score function can be extended to a larger set by patching scores on triplets.

By the symmetry it is now clear that the pivoting condition of $u$ and $w$ with respect to $v$ implies the pivoting condition of $u$ and $v$ with respect to $w$ and of $v$ and $w$ with respect to $u$. In other words, the pivoting condition is a property of the triplet $\{u, v, w\}$.

**Definition 3.3** *Assume a ranking model $\mathcal{D}$ for $U$ satisfies pairwise stability, and let $\Delta_{\mathcal{D}} : U \times U \to \mathbb{R}$ denote the comparison logit as defined above. A triplet $\{u, v, w\} \subseteq U$ is said to satisfy the* pivot condition *in $\mathcal{D}$ if $\Delta_{\mathcal{D}}(u, v) + \Delta_{\mathcal{D}}(v, w) + \Delta_{\mathcal{D}}(w, u) = 0$. We say that $U$ satisfies the pivot condition in $\mathcal{D}$ if $\{u, v, w\}$ satisfies the pivot condition for all $\{u, v, w\} \subseteq U$.*

**Lemma 3.1** *If $U$ satisfies the pivot condition in a pairwise stability model $\mathcal{D}$ for $U$, then there exists a real valued score function $s : V \to \mathbb{R}$ such that for all $a, b \in V$, $\Delta_{\mathcal{D}}(a, b) = s(a) - s(b)$ .*

**Proof** Fix some element $v \in U$ and set $s(v) = 0$. For every other element $u \in V \setminus \{v\}$ set $s(v) = \Delta_{\mathcal{D}}(v, u)$. It is now immediate to verify that for all $a, b \subseteq V$ one has $\Delta_{\mathcal{D}}(a, b) = s(a) - s(b)$. Indeed, by construction $s(a) - s(b) = \Delta_{\mathcal{D}}(a, u) - \Delta_{\mathcal{D}}(b, u)$ but by the pivot property this equals exactly $\Delta_D(a, b)$, as required (remember that $\Delta_{\mathcal{D}}(a, b) = -\Delta_{\mathcal{D}}(a, b)$ by definition of $\Delta_{\mathcal{D}}$).

By starting with *local* assumptions (pairwise stability and the pivoting property), we obtained a natural *global* score function $s$ on the universe of elements. The score function governs the probability of $u$ preceding $v$ via the difference $s(u) - s(v)$ passed through the inverse logit. Note that we used the assumption that the comparison logit is finite on all $u, v$ (equivalently, that $0 < \Pr_{\mathcal{D}}(u \prec v) < 1$ for all $u, v$), but this assumption can be dropped if we allow the score function to obtain values in $\mathbb{R} + \omega\mathbb{Z}$, where $\omega$ is the limit ordinal of $\mathbb{R}$.

The Plackett-Luce model satisfies both pairwise stability and the pivot condition with $s(u) = \log \alpha(u)$. Hence our definitions are non empty. Inspired by recent work on the QuickSort algorithm [24] as a random process [4, 3, 5, 37], we define a new *symmetric* model based on a series of comparisons rather than choices from sets.

## 4   The New Ranking Model

We define a model called $\mathrm{QS}_s$ (short for QuickSort), parametrized by a score function $s : U \mapsto \mathbb{R}$ as follows. Given a finite subset $V \subset U$:

    1. Pick a "pivot element" $v$ uniformly at random from $V$.

2. For all $u \in V \setminus \{v\}$, place $u$ to the left of $v$ with probability $1/(1 + e^{s(v)-s(u)})$, and to the right with the remaining probability $1/(1 + e^{s(u)-s(v)})$, independently of all other choices.

3. Recurse on the left and on the right sides, and output the ranking of $V$ obtained by joining the results in an obvious way (left $\prec$ pivot $\prec$ right).

(The function $1/(1 + e^{-x})$ is the inverse logit function.) We shall soon see that QuickSort gives us back all the desired statistical local properties of a ranking models. That the model $\mathrm{QS}_s$ can be sampled efficiently is a simple consequence of the fact that QuickSort runs in expected time $O(n \log n)$ (some attention needs to be paid the fact that unlike in the textbook proofs for QuickSort the pivoting process is randomized, but this is not difficult [5]).

**Theorem 4.1** *The ranking model* $\mathrm{QS}_s$ *for $U$ satisfies both pairwise stability and the pivoting condition. Additionally, for any subset $V \subseteq U$ the mode of $\mathrm{QS}_s(V)$ is any ranking $\pi^*$ satisfying $u \prec_{\pi^*} v$ whenever $s(u) > s(v)$.*

**Proof (of Theorem 4.1):** First we note that if $\mathrm{QS}_s$ satisfies pairwise stability, then the pivot property will be implied as well. Indeed, by taking $V = \{u, v\}$ we would get from the model that $\mathrm{Pr}_{\mathrm{QS}_s}(u \prec v) = 1/(1 + e^{s(v)-s(u)})$, immediately implying the pivot property.

To see that $\mathrm{QS}_s$ satisfies pairwise stability, we show that for any $u, v$ and $V \supseteq \{u, v\}$, the probability of the event $u \prec_\pi v$ is exactly $1/(1 + e^{s(v)-s(u)})$, where $\pi \sim \mathrm{QS}_s(V)$. Indeed, the order of $u, v$ can be determined in one of two ways. (i) *Directly*: $u$ or $v$ are chosen as pivot when the other is present in the same recursive call. We call this event $E_{\{u,v\}}$. Conditioned on this event, clearly the probability that $u \prec_\pi v$ is exactly the required probability $1/(1 + e^{s(v)-s(u)})$ by step 2 of QuickSort (note that it doesn't matter which one of $v$ or $u$ is the pivot). (ii) *Indirectly*: A third element $w \in V$ is the pivot when both $u$ and $v$ are present in the recursive call, and $w$ sends $u$ and $v$ to opposite recursion sides. We denote this event by $E'_{\{u,v\},w}$. *Conditioned* on this event, the probability that $u \prec_\pi v$, is exactly as required (by using the same logit calculus we used in Section 3).

To conclude the proof of pairwise stability, it remains to observe that the collection of events $\left\{ E_{\{u,v\}} \right\} \cup \left\{ E'_{\{u,v\},w} : w \in V \setminus \{u,v\} \right\}$ is a pairwise disjoint cover of the probability space. This implies that $\mathrm{Pr}_{\pi \sim \mathrm{QS}_s(V)}(u \prec_\pi v)$ is the desired quantity $1/(1 + e^{s(v)-s(u)})$, concluding the proof of pairwise stability.

We need to work harder to prove the intuitive mode argument. Let $\tau, \sigma$ be two permutations on $V$ such that

$$a_1 \prec_\tau a_2 \prec_\tau \cdots \prec_\tau a_k \prec_\tau u \prec_\tau v \prec_\tau a_{k+1} \prec_\tau \cdots \prec_\tau a_{n-2}$$

$$a_1 \prec a_2 \prec_\sigma \cdots \prec_\sigma a_k \prec_\sigma v \prec_\sigma u \prec_\sigma a_{k+1} \prec_\sigma \cdots \prec_\sigma a_{n-2} \, ,$$

where $V = \{u, v\} \cup \{a_1, \ldots, a_{n-2}\}$. In words, $\tau$ and $\sigma$ differ on the order of exactly two consecutive elements $u, v$. Assume that $s(u) > s(v)$ (so $\tau$, placing $u$ in a more favorable position than $v$, is intuitively more "correct"). We will prove that the probability of getting $\tau$ is strictly higher than the probability of getting $\sigma$ from $\mathrm{QS}_s$. Since $\pi^*$, the permutation sorting by $s$, can be obtained from any permutation by a sequence of swapping incorrectly ordered (according to $s$) adjacent pairs, this would prove the theorem by a standard inductive argument.

Let $q_\tau = \mathrm{Pr}_{\pi \sim \mathrm{QS}}[\pi = \tau]$, and similarly define $q_\sigma$. To prove that $q_\tau > q_\sigma$ we need extra notation. Our QuickSort generative model gives rise to a random integer node-labeled ordered binary tree[4] implicitly constructed as an execution side effect. This tree records the final position of the pivots chosen in each step as follows: The label $L$ of the root of the tree is the rank of the pivot in the final solution (which equals the size of the left recursion plus 1). The left subtree is the tree recursively constructed on the left, and the right subtree is the tree recursively constructed on the right with L added to the labels of all the vertices. Clearly the resulting tree has exactly $n$ nodes with each label in $\{1 \ldots n\}$ appearing exactly once. Let $p_{\pi,T}$ denote the probability that QuickSort outputs a permutation $\pi$ and (implicitly) constructs a pivot selection tree $T$. Let $\mathcal{T}$ denote the collection of all ordered labeled binary trees with node labels in $\{1, \ldots, n\}$. For $T \in \mathcal{T}$ and a node $x \in T$ let $\ell(x)$ denote the integer label on $x$. Let $T_x$ denote the subtree rooted by $x$ and let $\ell(T_x)$ denote the

collection of labels on those nodes. By construction, if QuickSort outputted a ranking $\pi$ with an (implicitly constructed) tree $T$, then at some point the recursive call to QuickSort took $\pi^{-1}(\ell(T_x))$ as input and chose $\pi^{-1}(\ell(x))$ as pivot, for any node $x$ of $T$. By a standard probability argument (summing over a disjoint cover of events): $q_\tau = \sum_{T \in \mathcal{T}} q_{\tau,T}$ and $q_\sigma = \sum_{T \in \mathcal{T}} q_{\sigma,T}$ . It suffices to show now that for any fixed $T \in \mathcal{T}$, $q_{\tau,T} > q_{\sigma,T}$. To compute $q_{\pi,T}$ for $\pi = \tau, \sigma$ we proceed as follows: At each node $x$ of $T$ we will attach a number $P_\pi(x)$ which is the likelihood of the decisions made at that level, namely, the choice of the pivot itself and the separation of the rest of the elements to its right and left.

$$P_\pi(x) = \frac{1}{|T_x|} \prod_{y \in T_{\mathrm{L}}(x)} \Pr_{\mathrm{QS}}[\pi^{-1}(\ell(y)) \prec \pi^{-1}(\ell(x))] \; \times \prod_{y \in T_{\mathrm{R}}(x)} \Pr_{\mathrm{QS}}[\pi^{-1}(\ell(x)) \prec \pi^{-1}(\ell(y))] \,,$$

Where $|T_x|$ is the number of nodes in $T_x$, $T_{\mathrm{R}}(x)$ is the set of vertices in the left subtree of $x$ and similarly for $T_{\mathrm{L}}(x)$. The factor $1/|T_x|$ comes from the likelihood of uniformly at random having chosen the pivot $\pi^{-1}(\ell(x))$ from the set of nodes of $T_x$. The first product corresponds to the random comparison decisions made on the elements thrown to the left, and the second to right. By construction, $p_{\tau,T} = \prod_{x \in T} P_\tau(x)$ and similarly $p_{\sigma,T} = \prod_{x \in T} P_\sigma(x)$. Since $u, v$ are adjacent in both $\tau$ and $\sigma$, it is clear that the two nodes $x_1, x_2 \in T$ labeled $\tau(u)$ and $\tau(v)$ respectively have an ancestor-descendent relation in $T$ (otherwise their least common ancestor in $T$ would have been placed between them, violating the consecutiveness of $u$ and $v$ in our construction and implying $p_{\tau,T} = q_{\tau,T} = 0$). Also recall that $\sigma(u) = \tau(v)$ and $\sigma(v) = \tau(u)$. By our assumption that $\tau$ and $\sigma$ differ only on the order of the adjacent elements $u, v$, $P_\tau(x)$ and $P_\sigma(x)$ could differ only on nodes $x$ on the path between $x_1$ and $x_2$. Assume w.l.o.g. that $x_1$ is an ancestor of $x_2$, and that $x_2$ is a node in the left subtree of $x_1$. By our construction, $x_2$ is the rightmost node[5] in $T_{\mathrm{L}}(x_1)$. Let $Y$ denote the set of nodes on the path from $x_1$ to $x_2$ (exclusive) in $T$. Let $W$ denote the set of nodes in the left (and only) subtree of $x_2$, and let $Z$ denote the set of remaining nodes in $T_{\mathrm{L}}(x_1)$: $Z = T_{\mathrm{L}}(x_1) \setminus (W \cup Y \cup \{x_2\})$. Since $\tau^{-1}(\ell(z)) = \sigma^{-1}(\ell(z))$ for all $z \in Z$ we can define $\mathrm{elt}(z) = \tau^{-1}(\ell(z)) = \sigma^{-1}(\ell(z))$ and similarly we can correspond each $y \in Y$ with a single element $\mathrm{elt}(y)$ and each $w \in W$ with a single elements $\mathrm{elt}(w)$ of $V$. As claimed above, we only need to compare between $P_\tau(x_1)$ and $P_\sigma(x_1)$, between $P_\tau(x_2)$ and $P_\sigma(x_2)$ and $P_\tau(y)$ and $P_\sigma(y)$ for $y \in Y$. Carefully unfolding these products node by node, we see that it suffices to notice that for all $y \in Y$, the probability of throwing $\mathrm{elt}(y)$ to the left of $u$ (pivoting on $u$) times the probability of throwing $v$ to the right of $\mathrm{elt}(y)$ (pivoting on $\mathrm{elt}(y)$) as appears inside the product $P_\sigma(x_1)P_\sigma(y)$ is exactly the probability of throwing $\mathrm{elt}(y)$ to the left of $v$ (pivoting on $v$) times the probability of throwing $u$ to the right of $\mathrm{elt}(y)$ (pivoting on $\mathrm{elt}(y)$) as appears inside the product $P_\tau(x_1)P_\tau(y)$. Also for all $w \in W$ the probability of throwing $\mathrm{elt}(w)$ to the left of $u$ (pivoting on $u$) times the probability of throwing $\mathrm{elt}(w)$ to the left of $v$ (pivoting on $v$) appears exactly once in both $P_\tau(x_1)P_\tau(x_2)$ and $P_\sigma(x_1)P_\sigma(x_2)$ (though in reversed order). Following these observations one can be convinced by the desired result of the theorem by noting that in virtue of $s(u) > s(v)$: (i) $\Pr_{\mathrm{QS}}[v \prec u] > \Pr_{\mathrm{QS}}[u \prec v]$, and (ii) for all $z \in Z$, $\Pr_{\mathrm{QS}}[\mathrm{elt}(z) \prec u] > \Pr_{\mathrm{QS}}[\mathrm{elt}(z) \prec v]$.

## 5 Comparison of Models

The stochastic QuickSort model as just defined as well as Plackett-Luce share much in common, but they are not identical for strictly more than 2 elements. Both satisfy the intuitive property that the mode of the distribution corresponding to a set $V$ is any ranking which sorts the elements of $V$ in decreasing $s(v) = \log \alpha(v)$ value. The stochastic QuickSort model, however, does not suffer from the asymmetry problem which is often stated as a criticism of Plackett-Luce. Indeed, the distributions $\mathrm{QS}_s(V)$ has the following property: If we draw from $\mathrm{QS}_s(V)$ and flip the resulting permutation, the resulting distribution is $\mathrm{QS}_{-s}(V)$. This property does not hold in general for Plackett-Luce, and hence serves as proof of their nonequivalence.

Assume we want to fit $s$ in the MLE sense by drawing random permutations from $\mathrm{QS}_s(V)$. This seems to be difficult due to the unknown choice of pivot. On the other hand, the log-likelihood function corresponding to Plackett-Luce is globally concave in the values of the function $s$ on $V$, and hence a global maximum can be efficiently found. This also holds true in a generalized linear model, in which $s(v)$ is given as the dot product of a feature vector $\phi(v)$ with an unknown weight

vector which we estimate (as done in [10] in the context of predicting demand for electric cars). Hence, for the purpose of learning given full permutations of strictly more than two elements, the Plackett-Luce model is easier to work with.

In practical IR settings, however, it is rare that training data is obtained as full permutations: such a task is tiresome. In most applications, the observables used for training are in the form of binary response vectors (either *relevant* or *irrelevant* for each alternative) or comparison of pairs of alternatives (either *A better* or *B better* given A,B). For the latter, Plackett-Luce is identical to Quick-Sort, and hence efficient fitting of parameters is easy (using logistic regression). As for the former, the process of generating a binary response vector can be viewed as the task performed at a single QuickSort recursive level. It turns out that by defining a nuisance parameter to represent the value $s$ of an unknown pivot, MLE estimation can be performed efficiently and exactly [2].

## Footnotes

[1]This choice function is known as the multinomial logit (MNL) and is equivalent to the standard (dichotomous) logit when only two alternatives are available.

[2]We choose in this work to use the convention that an element $u$ precedes $v$ if $u$ is in a *more favorable* position. When a score function is introduced later, the convention will be that higher scores correspond to more favorable positions. We will use the symbol $<$ (resp. $>$) to compare scores, which is semantically opposite to $\prec$ (resp. $\succ$) by our convention.

[3]The "logit of $p$" is standard shorthand for the log-odds, or $\log(p/(1 - p))$.

[4]By that we mean a tree in which each node has at most one *left* child node and at most one *right* child node, and the nodes are labeled with integers.

[5]The rightmost node of $T$ is the root if it has no right descendent, or the rightmost node of its right subtree.

## References

[1] Shivani Agarwal and Partha Niyogi. Stability and generalization of bipartite ranking algorithms. In *COLT*, pages 32–47, 2005.

[2] N. Ailon. A simple linear ranking algorithm using query dependent intercept variables. `arXiv:0810.2764v1`.

[3] Nir Ailon. Aggregation of partial rankings, p-ratings and top-m lists. In *SODA*, 2007.

[4] Nir Ailon, Moses Charikar, and Alantha Newman. Aggregating inconsistent information: ranking and clustering. In *Proceedings of the 37th Annual ACM Symposium on Theory of Computing, Baltimore, MD, USA, May 22-24, 2005*, pages 684–693. ACM, 2005.

[5] Nir Ailon and Mehryar Mohri. An efficient reduction of ranking to classification. In *COLT*, 2008.

[6] Erin L. Allwein, Robert E. Schapire, and Yoram Singer. Reducing multiclass to binary: A unifying approach for margin classifiers. *Journal of Machine Learning Research*, 1:113–141, 2000.

[7] D. Ariely, G. Loewenstein, and D. Prelec. Coherent arbitrariness: Stable demand curves without stable preferences. *The Quarterly Journal of Economics*, 118(1):73–105, 2008.

[8] K. J. Arrow. A difficulty in the concept of social welfare. *Journal of Political Economy*, 58(4):328–346, August 1950.

[9] Maria-Florina Balcan, Nikhil Bansal, Alina Beygelzimer, Don Coppersmith, John Langford, and Gregory B. Sorkin. Robust reductions from ranking to classification. In Nader H. Bshouty and Claudio Gentile, editors, *COLT*, volume 4539 of *Lecture Notes in Computer Science*, pages 604–619. Springer, 2007.

[10] S. Beggs and S. Cardell. Assessing the potential demand for electric cars. *Journal of Econometrics*, 17:1–19, 1981.

[11] R.A. Bradley and M.A. Terry. Rank analysis of incomplete block designs. *Biometrika*, 39:324–345, 1952.

[12] Zhe Cao, Tao Qin, Tie-Yan Liu, Ming-Feng Tsai, and Hang Li. Learning to rank: from pairwise approach to listwise approach. In *ICML '07: Proceedings of the 24th international conference on Machine learning*, pages 129–136, New York, NY, USA, 2007. ACM.

[13] William W. Cohen, Robert E. Schapire, and Yoram Singer. Learning to order things. *J. Artif. Intell. Res. (JAIR)*, 10:243–270, 1999.

[14] D. Coppersmith, Lisa Fleischer, and Atri Rudra. Ordering by weighted number of wins gives a good ranking for weighted tournamnets. In *Proceedings of the 17th Annual ACM-SIAM Symposium on Discrete Algorithms (SODA)*, 2006.

[15] Corinna Cortes and Mehryar Mohri. AUC Optimization vs. Error Rate Minimization. In *Advances in Neural Information Processing Systems (NIPS 2003)*, volume 16, Vancouver, Canada, 2004. MIT Press.

[16] Corinna Cortes, Mehryar Mohri, and Ashish Rastogi. An Alternative Ranking Problem for Search Engines. In *Proceedings of the 6th Workshop on Experimental Algorithms (WEA 2007)*, volume 4525 of *Lecture Notes in Computer Science*, pages 1–21, Rome, Italy, June 2007. Springer-Verlag, Heidelberg, Germany.

[17] Corinna Cortes, Mehryar Mohri, and Ashish Rastogi. Magnitude-Preserving Ranking Algorithms. In *Proceedings of the Twenty-fourth International Conference on Machine Learning (ICML 2007)*, Oregon State University, Corvallis, OR, June 2007.

[18] David Cossock and Tong Zhang. Subset ranking using regression. In *COLT*, pages 605–619, 2006.

[19] Koby Crammer and Yoram Singer. Pranking with ranking. In Thomas G. Dietterich, Suzanna Becker, and Zoubin Ghahramani, editors, *Advances in Neural Information Processing Systems 14 [Neural Information Processing Systems: Natural and Synthetic, NIPS 2001, December 3-8, 2001, Vancouver, British Columbia, Canada]*, pages 641–647. MIT Press, 2001.

[20] Ronald Fagin, Ravi Kumar, Mohammad Mahdian, D. Sivakumar, and Erik Vee. Comparing and aggregating rankings with ties. In Alin Deutsch, editor, *Proceedings of the Twenty-third ACM SIGACT-SIGMOD-SIGART Symposium on Principles of Database Systems, June 14-16, 2004, Paris, France*, pages 47–58. ACM, 2004.

[21] Yoav Freund, Raj D. Iyer, Robert E. Schapire, and Yoram Singer. An efficient boosting algorithm for combining preferences. *Journal of Machine Learning Research*, 4:933–969, 2003.

[22] Ralf Herbrich, Thore Graepel, Peter Bollmann-Sdorra, and Klaus Obermayer. Learning a preference relation in ir. In *In Proceedings Workshop Text Categorization and Machine Learning, International Conference on Machine Learning*, pages 80–84, 1998.

[23] Ralf Herbrich, Thore Graepel, and Klaus Obermayer. Large margin rank boundaries for ordinal regression. In *Advances in Large Margin Classifiers*, pages 115–132, 2000.

[24] C.A.R. Hoare. Quicksort: Algorithm 64. *Comm. ACM*, 4(7):321–322, 1961.

[25] Thorsten Joachims. Optimizing search engines using clickthrough data. In *KDD '02: Proceedings of the eighth ACM SIGKDD international conference on Knowledge discovery and data mining*, pages 133–142, New York, NY, USA, 2002. ACM Press.

[26] Claire Kenyon-Mathieu and Warren Schudy. How to rank with few errors. In *STOC '07: Proceedings of the thirty-ninth annual ACM symposium on Theory of computing*, pages 95–103, New York, NY, USA, 2007. ACM Press.

[27] Guy Lebanon and John D. Lafferty. Cranking: Combining rankings using conditional probability models on permutations. In *ICML '02: Proceedings of the Nineteenth International Conference on Machine Learning*, pages 363–370, San Francisco, CA, USA, 2002. Morgan Kaufmann Publishers Inc.

[28] Erich L. Lehmann. *Nonparametrics: Statistical Methods Based on Ranks*. Holden-Day, San Francisco, California, 1975.

[29] R.D. Luce. *Individual choice behaviour*. Wiley, 1959.

[30] C.L. Mallows. Non-null ranking models. *Biometrika*, 44:113–130, 1957.

[31] John I. Marden. *Analyzing and modeling rank data*. Chapman & Hall, 1995.

[32] P. McCullagh. Permutations and regression models. *Probability models and statistical analyses for ranking data*, pages 196–215, 1993.

[33] Mark H. Montague and Javed A. Aslam. Condorcet fusion for improved retrieval. In *Proceedings of the 2002 ACM CIKM International Conference on Information and Knowledge Management, McLean, VA, USA, November 4-9, 2002*, pages 538–548. ACM, 2002.

[34] R. L. Plackett. The analysis of permutations. *Applied Statistics*, 24:193–202.

[35] Cynthia Rudin, Corinna Cortes, Mehryar Mohri, and Robert E. Schapire. Margin-based ranking meets boosting in the middle. In Peter Auer and Ron Meir, editors, *Learning Theory, 18th Annual Conference on Learning Theory, COLT 2005, Bertinoro, Italy, June 27-30, 2005, Proceedings*, pages 63–78. Springer, 2005.

[36] L. L. Thurstone. A law of comparative judgement. *Psychological Reviews*, 34:273–286.

[37] David P. Williamson and Anke van Zuylen. "deterministic algorithms for rank aggregation and other ranking and clustering problems". In *Proceedings of the 5th Workshop on Approximation and Online Algorithms (WAOA) (to appear)*, 2007.

[38] J. Yellott. The relationship between luce's choice axiom, thurstone's theory of comparatice judgment, and the double exponential distribution. *Journal of Mathematical Psychology*, 15:109–144, 1977.
